# Inverse Reinforcement Learning through Structured Classification

**Edouard Klein**[1,2]
[1]LORIA – team ABC
Nancy, France
edouard.klein@supelec.fr

**Matthieu Geist**[2]
[2]Supélec – IMS-MaLIS Research Group
Metz, France
matthieu.geist@supelec.fr

**Bilal Piot**[2,3]**, Olivier Pietquin**[2,3]
[3] UMI 2958 (GeorgiaTech-CNRS)
Metz, France
{bilal.piot,olivier.pietquin}@supelec.fr

## Abstract

This paper adresses the inverse reinforcement learning (IRL) problem, that is inferring a reward for which a demonstrated expert behavior is optimal. We introduce a new algorithm, SCIRL, whose principle is to use the so-called feature expectation of the expert as the parameterization of the score function of a multi-class classifier. This approach produces a reward function for which the expert policy is provably near-optimal. Contrary to most of existing IRL algorithms, SCIRL does not require solving the direct RL problem. Moreover, with an appropriate heuristic, it can succeed with only trajectories sampled according to the expert behavior. This is illustrated on a car driving simulator.

## 1 Introduction

Inverse reinforcement learning (IRL) [14] consists in finding a reward function such that a demonstrated expert behavior is optimal. Many IRL algorithms (to be briefly reviewed in Sec. 5) search for a reward function such that the associated optimal policy induces a distribution over trajectories (or some measure of this distribution) which matches the one induced by the expert. Often, this distribution is characterized by the so-called feature expectation (see Sec. 2.1): given a reward function linearly parameterized by some feature vector, it is the expected discounted cumulative feature vector for starting in a given state, applying a given action and following the related policy.

In this paper, we take a different route. The expert behavior could be mimicked by a supervised learning algorithm generalizing the mapping from states to actions. Here, we consider generally multi-class classifiers which compute from a training set the parameters of a linearly parameterized score function; the decision rule for a given state is the argument (the action) which maximizes the score function for this state (see Sec. 2.2). The basic idea of our SCIRL (Structured Classification-based IRL) algorithm is simply to take an estimate of the expert feature expectation as the parameterization of the score function (see Sec. 3.1). The computed parameter vector actually defines a reward function for which we show the expert policy to be near-optimal (Sec. 3.2).

Contrary to most existing IRL algorithms, a clear advantage of SCIRL is that it does not require solving repeatedly the direct reinforcement learning (RL) problem. It requires estimating the expert feature expectation, but this is roughly a policy evaluation problem (for an observed policy, so is less involved than repeated policy optimization problems), see Sec. 4. Moreover, up to the use of some heuristic, SCIRL may be trained solely from transitions sampled from the expert policy (no need to sample the whole dynamic). We illustrate this on a car driving simulator in Sec. 6.

## 2 Background and Notations

### 2.1 (Inverse) Reinforcement Learning

A Markov Decision process (MDP) [12] is a tuple $\{\mathcal{S}, \mathcal{A}, \mathcal{P}, \mathcal{R}, \gamma\}$ where $\mathcal{S}$ is the finite state space[1], $\mathcal{A}$ the finite actions space, $\mathcal{P} = \{P_a = (p(s'|s,a))_{1 \le s,s' \le |\mathcal{S}|}, a \in \mathcal{A}\}$ the set of Markovian transition probabilities, $\mathcal{R} \in \mathbb{R}^{\mathcal{S}}$ the state-dependent reward function and $\gamma$ the discount factor. A deterministic policy $\pi \in \mathcal{S}^{\mathcal{A}}$ defines the behavior of an agent. The quality of this control is quantified by the value function $v_{\mathcal{R}}^{\pi} \in \mathbb{R}^{\mathcal{S}}$, associating to each state the cumulative discounted reward for starting in this state and following the policy $\pi$ afterwards: $v_{\mathcal{R}}^{\pi}(s) = \mathbb{E}[\sum_{t \ge 0} \gamma^t \mathcal{R}(S_t)|S_0 = s, \pi]$. An optimal policy $\pi_{\mathcal{R}}^*$ (according to the reward function $\mathcal{R}$) is a policy of associated value function $v_{\mathcal{R}}^*$ satisfying $v_{\mathcal{R}}^* \ge v_{\mathcal{R}}^{\pi}$, for any policy $\pi$ and componentwise.

Let $P_{\pi}$ be the stochastic matrix $P_{\pi} = (p(s'|s, \pi(s)))_{1 \le s,s' \le |\mathcal{S}|}$. With a slight abuse of notation, we may write $a$ the policy which associates the action $a$ to each state $s$. The Bellman evaluation (resp. optimality) operators $T_{\mathcal{R}}^{\pi}$ (resp. $T_{\mathcal{R}}^*$) : $\mathbb{R}^{\mathcal{S}} \rightarrow \mathbb{R}^{\mathcal{S}}$ are defined as $T_{\mathcal{R}}^{\pi} v = \mathcal{R} + \gamma P_{\pi} v$ and $T_{\mathcal{R}}^* v = \max_{\pi} T_{\mathcal{R}}^{\pi} v$. These operators are contractions and $v_{\mathcal{R}}^{\pi}$ and $v_{\mathcal{R}}^*$ are their respective fixed-points: $v_{\mathcal{R}}^{\pi} = T_{\mathcal{R}}^{\pi} v_{\mathcal{R}}^{\pi}$ and $v_{\mathcal{R}}^* = T_{\mathcal{R}}^* v_{\mathcal{R}}^*$. The action-value function $Q^{\pi} \in \mathbb{R}^{\mathcal{S} \times \mathcal{A}}$ adds a degree of freedom on the choice of the first action, it is formally defined as $Q_{\mathcal{R}}^{\pi}(s,a) = [T_{\mathcal{R}}^a v_{\mathcal{R}}^{\pi}](s)$. We also write $\rho_{\pi}$ the stationary distribution of the policy $\pi$ (satisfying $\rho_{\pi}^{\top} P_{\pi} = \rho_{\pi}^{\top}$).

Reinforcement learning and approximate dynamic programming aim at estimating the optimal control policy $\pi_{\mathcal{R}}^*$ when the model (transition probabilities and the reward function) is unknown (but observed through interactions with the system to be controlled) and when the state space is too large to allow exact representations of the objects of interest (as value functions or policies) [2, 15, 17]. We refer to this as the direct problem. On the contrary, (approximate) inverse reinforcement learning [11] aim at estimating a reward function for which an observed policy is (nearly) optimal. Let us call this policy the expert policy, denoted $\pi_E$. We may assume that it optimizes some unknown reward function $\mathcal{R}_E$. The aim of IRL is to compute some reward $\hat{\mathcal{R}}$ such that the expert policy is (close to be) optimal, that is such that $v_{\hat{\mathcal{R}}}^* \approx v_{\hat{\mathcal{R}}}^{\pi_E}$. We refer to this as the inverse problem.

Similarly to the direct problem, the state space may be too large for the reward function to admit a practical exact representation. Therefore, we restrict our search of a good reward among linearly parameterized functions. Let $\phi(s) = (\phi_1(s) \ldots \phi_p(s))^{\top}$ be a feature vector composed of $p$ basis function $\phi_i \in \mathbb{R}^{\mathcal{S}}$, we define the parameterized reward functions as $\mathcal{R}_{\theta}(s) = \theta^{\top} \phi(s) = \sum_{i=1}^{p} \theta_i \phi_i(s)$. Searching a good reward thus reduces to searching a good parameter vector $\theta \in \mathbb{R}^p$. Notice that we will use interchangeably $\mathcal{R}_{\theta}$ and $\theta$ as subscripts (e.g., $v_{\theta}^{\pi}$ for $v_{\mathcal{R}_{\theta}}^{\pi}$). Parameterizing the reward this way implies a related parameterization for the action-value function:

$$Q_{\theta}^{\pi}(s,a) = \theta^{\top} \mu^{\pi}(s,a) \text{ with } \mu^{\pi}(s,a) = \mathbb{E}[\sum_{t \ge 0} \gamma^t \phi(S_t)|S_0 = s, A_0 = a, \pi]. \qquad (1)$$

Therefore, the action-value function shares the parameter vector of the reward function, with an associated feature vector $\mu^{\pi}$ called the *feature expectation*. This notion will be of primary importance for the contribution of this paper. Notice that each component $\mu_i^{\pi}$ of this feature vector is actually the action-value function of the policy $\pi$ assuming the reward is $\phi_i$: $\mu_i^{\pi}(s,a) = Q_{\phi_i}^{\pi}(s,a)$. Therefore, any algorithm designed for estimating an action-value function may be used to estimate the feature expectation, such as Monte-Carlo rollouts or temporal difference learning [7].

### 2.2 Classification with Linearly Parameterized Score Functions

Let $\mathcal{X}$ be a compact or a finite set (of inputs to be classified) and let $\mathcal{Y}$ be a finite set (of labels). Assume that inputs $x \in \mathcal{X}$ are drawn according to some unknown distribution $\mathbb{P}(x)$ and that there exists some oracle which associates to each of these inputs a label $y \in Y$ drawn according to the unknown conditional distribution $\mathbb{P}(y|x)$. Generally speaking, the goal of multi-class classification is, given a training set $\{(x_i, y_i)_{1 \le i \le N}\}$ drawn according to $\mathbb{P}(x,y)$, to produce a decision rule $g \in \mathcal{Y}^{\mathcal{X}}$ which aims at minimizing the classification error $\mathbb{E}[\chi_{\{g(x) \ne y\}}] = \mathbb{P}(g(x) \ne y)$, where $\chi$ denotes the indicator function.

Here, we consider a more restrictive set of classification algorithms. We assume that the decision rule associates to an input the argument which maximizes a related score function, this score function being linearly parameterized and the associated parameters being learnt by the algorithm. More formally, let $\psi(s,a) = (\psi_1(x,y)\ldots\psi_d(x,y))^\top \in \mathbb{R}^d$ be a feature vector whose components are $d$ basis functions $\psi_i \in \mathbb{R}^{\mathcal{X}\times\mathcal{Y}}$. The linearly parameterized score function $s_w \in \mathbb{R}^{\mathcal{X}\times\mathcal{Y}}$ of parameter vector $w \in \mathbb{R}^d$ is defined as $s_w(x,y) = w^\top \psi(x,y)$. The associated decision rule $g_w \in \mathcal{Y}^{\mathcal{X}}$ is defined as $g_w(x) \in \mathrm{argmax}_{y\in\mathcal{Y}} s_w(x,y)$. Using a training set $\{(x_i, y_i)_{1\leq i\leq N}\}$, a linearly parameterized score function-based multi-class classification (MC$^2$ for short) algorithm computes a parameter vector $\theta_c$. The quality of the solution is quantified by the classification error $\epsilon_c = \mathbb{P}(g_{\theta_c}(x) \neq y)$.

We do not consider a specific MC$^2$ algorithm, as long as it classifies inputs by maximizing the argument of a linearly parameterized score function. For example, one may choose a multi-class support vector machine [6] (taking the kernel induced by the feature vector) or a structured large margin approach [18]. Other choices may be possible, one can choose its preferred algorithm.

## 3 Structured Classification for Inverse Reinforcement Learning

### 3.1 General Algorithm

Consider the classification framework of Sec. 2.2. The input $x$ may be seen as a state and the label $y$ as an action. Then, the decision rule $g_w(x)$ can be interpreted as a policy which is greedy according to the score function $w^\top\psi(x,y)$, which may itself be seen as an action-value function. Making the parallel with Eq. (1), if $\psi(x,y)$ is the feature expectation of some policy $\pi$ which produces labels of the training set, and if the classification error is small, then $w$ will be the parameter vector of a reward function for which we may hope the policy $\pi$ to be near optimal. Based on these remarks, we're ready to present the proposed Structured Classification-based IRL (SCIRL) algorithm.

Let $\pi_E$ be the expert policy from which we would like to recover a reward function. Assume that we have a training set $\mathcal{D} = \{(s_i, a_i = \pi_E(s_i))_{1\leq i\leq N}\}$ where states are sampled according to the expert stationary distribution[2] $\rho_E = \rho_{\pi_E}$. Assume also that we have an estimate $\hat{\mu}^{\pi_E}$ of the expert feature expectation $\mu^{\pi_E}$ defined in Eq. (1). How to practically estimate this quantity is postponed to Sec. 4.1; however, recall that estimating $\mu^{\pi_E}$ is simply a *policy evaluation* problem (estimating the action-value function of a given policy), as noted in Sec. 2.1. Assume also that an MC$^2$ algorithm has been chosen. The proposed algorithm simply consists in choosing $\theta^\top\hat{\mu}^{\pi_E}(s,a)$ as the linearly parameterized score function, training the classifier on $\mathcal{D}$ which produces a parameter vector $\theta_c$, and outputting the reward function $\mathcal{R}_{\theta_c}(s) = \theta_c^\top\phi(s)$.

---

**Algorithm 1**: SCIRL algorithm

---

***Given*** a training set $\mathcal{D} = \{(s_i, a_i = \pi_E(s_i))_{1\leq i\leq N}\}$, an estimate $\hat{\mu}^{\pi_E}$ of the expert feature expectation $\mu^{\pi_E}$ and an MC$^2$ algorithm;

***Compute*** the parameter vector $\theta_c$ using the MC$^2$ algorithm fed with the training set $\mathcal{D}$ and considering the parameterized score function $\theta^\top\hat{\mu}^{\pi_E}(s,a)$;

***Output*** the reward function $\mathcal{R}_{\theta_c}(s) = \theta_c^\top\phi(s)$ ;

---

The proposed approach is summarized in Alg. 1. We call this Structured Classification-based IRL because using the (estimated) expert feature expectation as the feature vector for the classifier somehow implies taking into account the MDP structure into the classification problem and allows outputting a reward vector. Notice that contrary to most of existing IRL algorithms, SCIRL does not require solving the direct problem. If it possibly requires estimating the expert feature expectation, it is just a policy evaluation problem, less difficult than the policy optimization issue involved by the direct problem. This is further discussed in Sec. 5.

## 3.2 Analysis

In this section, we show that the expert policy $\pi_E$ is close to be optimal according to the reward function $\mathcal{R}_{\theta_c}$, more precisely that $\mathbb{E}_{s \sim \rho_E}[v^*_{\theta_c}(s) - v^{\pi_E}_{\theta_c}(s)]$ is small. Before stating our main result, we need to introduce some notations and to define some objects.

We will use the first order discounted future state distribution concentration coefficient $C_f$ [9]:

$$C_f = (1-\gamma) \sum_{t \geq 0} \gamma^t c(t) \text{ with } c(t) = \max_{\pi_1,\ldots,\pi_t, s \in \mathcal{S}} \frac{(\rho_E^\top P_{\pi_1} \ldots P_{\pi_t})(s)}{\rho_E(s)}.$$

We note $\pi_c$ the decision rule of the classifier: $\pi_c(s) \in \mathrm{argmax}_{a \in \mathcal{A}} \theta_c^\top \hat{\mu}^{\pi_E}(s, a)$. The classification error is therefore $\epsilon_c = \mathbb{E}_{s \sim \rho_E}[\chi_{\{\pi_c(s) \neq \pi_E(s)\}}] \in [0,1]$. We write $\hat{Q}^{\pi_E}_{\theta_c} = \theta_c^\top \hat{\mu}^{\pi_E}$ the score function computed from the training set $\mathcal{D}$ (which can be interpreted as an approximate action-value function). Let also $\epsilon_\mu = \hat{\mu}^{\pi_E} - \mu^{\pi_E} : \mathcal{S} \times \mathcal{A} \to \mathbb{R}^p$ be the feature expectation error. Consequently, we define the action-value function error as $\epsilon_Q = \hat{Q}^{\pi_E}_{\theta_c} - Q^{\pi_E}_{\theta_c} = \theta_c^\top (\hat{\mu}^{\pi_E} - \mu^{\pi_E}) = \theta_c^\top \epsilon_\mu : \mathcal{S} \times \mathcal{A} \to \mathbb{R}$. We finally define the mean delta-max action-value function error as $\bar{\epsilon}_Q = \mathbb{E}_{s \sim \rho_E}[\max_{a \in \mathcal{A}} \epsilon_Q(s,a) - \min_{a \in \mathcal{A}} \epsilon_Q(s,a)] \geq 0$.

**Theorem 1.** *Let $\mathcal{R}_{\theta_c}$ be the reward function outputted by Alg. 1. Let also the quantities $C_f$, $\epsilon_c$ and $\bar{\epsilon}_Q$ be defined as above. We have*

$$0 \leq \mathbb{E}_{s \sim \rho_E}[v^*_{\mathcal{R}_{\theta_c}} - v^{\pi_E}_{\mathcal{R}_{\theta_c}}] \leq \frac{C_f}{1-\gamma}\left(\bar{\epsilon}_Q + \epsilon_c \frac{2\gamma \|\mathcal{R}_{\theta_c}\|_\infty}{1-\gamma}\right).$$

*Proof.* As the proof only relies on the reward $\mathcal{R}_{\theta_c}$, we omit the related subscripts to keep the notations simple (*e.g.*, $v^\pi$ for $v^\pi_{\theta_c} = v^\pi_{\mathcal{R}_{\theta_c}}$ or $\mathcal{R}$ for $\mathcal{R}_{\theta_c}$). First, we link the error $\mathbb{E}_{s \sim \rho_E}[v^*(s) - v^{\pi_E}(s)]$ to the Bellman residual $\mathbb{E}_{s \sim \rho_E}[[T^* v^{\pi_E}](s) - v^{\pi_E}(s)]$. Componentwise, we have that:

$$v^* - v^{\pi_E} = T^* v^* - T^{\pi^*} v^{\pi_E} + T^{\pi^*} v^{\pi_E} - T^* v^{\pi_E} + T^* v^{\pi_E} - v^{\pi_E}$$

$$\overset{(a)}{\leq} \gamma P_{\pi^*}(v^* - v^{\pi_E}) + T^* v^{\pi_E} - v^{\pi_E} \overset{(b)}{\leq} (I - \gamma P_{\pi^*})^{-1}(T^* v^{\pi_E} - v^{\pi_E}).$$

Inequality $(a)$ holds because $T^{\pi^*} v^{\pi_E} \leq T^* v^{\pi_E}$ and inequality $(b)$ holds thanks to [9, Lemma 4.2]. Moreover, $v^*$ being optimal we have that $v^* - v^{\pi_E} \geq 0$ and $T^*$ being the Bellman optimality operator, we have $T^* v^{\pi_E} \geq T^{\pi_E} v^{\pi_E} = v^{\pi_E}$. Additionally, remark that $(I - \gamma P_{\pi^*})^{-1} = \sum_{t \geq 0} \gamma^t P_{\pi^*}^t$. Therefore, using the definition of the concentration coefficient $C_f$, we have that:

$$0 \leq \mathbb{E}_{s \sim \rho_E}[v^*(s) - v^{\pi_E}(s)] \leq \frac{C_f}{1-\gamma} \mathbb{E}_{s \sim \rho_E}\left[[T^* v^{\pi_E}](s) - v^{\pi_E}(s)\right]. \tag{2}$$

This results actually follows closely the one of [9, Theorem 4.2]. There remains to bound the Bellman residual $\mathbb{E}_{s \sim \rho_E}[[T^* v^{\pi_E}](s) - v^{\pi_E}(s)]$. Considering the following decomposition,

$$T^* v^{\pi_E} - v^{\pi_E} = T^* v^{\pi_E} - T^{\pi_c} v^{\pi_E} + T^{\pi_c} v^{\pi_E} - v^{\pi_E},$$

we will bound $\mathbb{E}_{s \sim \rho_E}[[T^* v^{\pi_E}](s) - [T^{\pi_c} v^{\pi_E}](s)]$ and $\mathbb{E}_{s \sim \rho_E}[[T^{\pi_c} v^{\pi_E}](s) - v^{\pi_E}(s)]$.

The policy $\pi_c$ (the decision rule of the classifier) is greedy with respect to $\hat{Q}^{\pi_E} = \theta_c^\top \hat{\mu}^{\pi_E}$. Therefore, for any state-action couple $(s,a) \in \mathcal{S} \times \mathcal{A}$ we have:

$$\hat{Q}^{\pi_E}(s, \pi_c(s)) \geq \hat{Q}^{\pi_E}(s,a) \Leftrightarrow Q^{\pi_E}(s,a) \leq Q^{\pi_E}(s, \pi_c(s)) + \epsilon_Q(s, \pi_c(s)) - \epsilon_Q(s,a).$$

By definition, $Q^{\pi_E}(s,a) = [T^a v^{\pi_E}](s)$ and $Q^{\pi_E}(s, \pi_c(s)) = [T^{\pi_c} v^{\pi_E}](s)$. Therefore, for $s \in \mathcal{S}$:

$$\forall a \in A, \ [T^a v^{\pi_E}](s) \leq [T^{\pi_c} v^{\pi_E}](s) + \epsilon_Q(s, \pi_c(s)) - \epsilon_Q(s,a)$$
$$\Rightarrow [T^* v^{\pi_E}](s) \leq [T^{\pi_c} v^{\pi_E}](s) + \max_{a \in \mathcal{A}} \epsilon_Q(s,a) - \min_{a \in \mathcal{A}} \epsilon_Q(s,a).$$

Taking the expectation according to $\rho_E$ and noticing that $T^* v^{\pi_E} \geq v^{\pi_E}$, we bound the first term:

$$0 \leq \mathbb{E}_{s \sim \rho_E}[[T^* v^{\pi_E}](s) - [T^{\pi_c} v^{\pi_E}](s)] \leq \bar{\epsilon}_Q. \tag{3}$$

There finally remains to bound the term $\mathbb{E}_{s \sim \rho_E}[[T^{\pi_c} v^{\pi_E}](s) - v^{\pi_E}(s)]$.

Let us write $M \in \mathbb{R}^{|\mathcal{S}| \times |\mathcal{S}|}$ the diagonal matrix defined as $M = \mathrm{diag}(\chi_{\{\pi_c(s) \neq \pi_E(s)\}})$. Using this, the Bellman operator $T^{\pi_c}$ may be written as, for any $v \in \mathbb{R}^{\mathcal{S}}$:

$$T^{\pi_c} v = \mathcal{R} + \gamma M P_{\pi_c} v + \gamma (I - M) P_{\pi_E} v = \mathcal{R} + \gamma P_{\pi_E} v + \gamma M (P_{\pi_c} - P_{\pi_E}) v.$$

Applying this operator to $v^{\pi_E}$ and recalling that $\mathcal{R} + \gamma P_{\pi_E} v^{\pi_E} = T^{\pi_E} v^{\pi_E} = v^{\pi_E}$, we get:

$$T^{\pi_c} v^{\pi_E} - v^{\pi_E} = \gamma M (P_{\pi_c} - P_{\pi_E}) v^{\pi_E} \Rightarrow |\rho_E^\top (T^{\pi_c} v^{\pi_E} - v^{\pi_E})| = \gamma |\rho_E^\top M (P_{\pi_c} - P_{\pi_E}) v^{\pi_E}|.$$

One can easily see that $\|(P_{\pi_c} - P_{\pi_E}) v^{\pi_E}\|_\infty \leq \frac{2}{1-\gamma} \|\mathcal{R}\|_\infty$, which allows bounding the last term:

$$|\mathbb{E}_{s \sim \rho_E}[[T^{\pi_c} v^{\pi_E}](s) - v^{\pi_E}(s)]| \leq \epsilon_c \frac{2\gamma}{1-\gamma} \|\mathcal{R}\|_\infty. \tag{4}$$

Injecting bounds of Eqs. (3) and (4) into Eq. (2) gives the stated result. □

This result shows that if the expert feature expectation is well estimated (in the sense that the estimation error $\epsilon_\mu$ is small for states sampled according to the expert stationary policy and for all actions) and if the classification error $\epsilon_c$ is small, then the proposed generic algorithm outputs a reward function $\mathcal{R}_{\theta_c}$ for which the expert policy will be near optimal. A direct corollary of Th. 1 is that given the true expert feature expectation $\mu^{\pi_E}$ and a perfect classifier ($\epsilon_c = 0$), $\pi_E$ is the unique optimal policy for $\mathcal{R}_{\theta_c}$.

One may argue that this bounds trivially holds for the null reward function (a reward often exhibited to show that IRL is an ill-posed problem), obtained if $\theta_c = 0$. However, recall that the parameter vector $\theta_c$ is computed by the classifier. With $\theta_c = 0$, the decision rule would be a random policy and we would have $\epsilon_c = \frac{|\mathcal{A}|-1}{|\mathcal{A}|}$, the worst possible classification error. This case is really unlikely. Therefore, we advocate that the proposed approach somehow allows disambiguating the IRL problem (at least, it does not output trivial reward functions such as the null vector). Also, this bound is scale-invariant: one could impose $\|\theta_c\| = 1$ or normalize (action-) value functions by $\|\mathcal{R}_{\theta_c}\|_\infty^{-1}$.

One should notice that there is a hidden dependency of the classification error $\epsilon_c$ to the estimated expert feature expectation $\hat{\mu}^{\pi_E}$. Indeed, the minimum classification error depends on the hypothesis space spanned by the chosen score function basis functions for the $\mathrm{MC}^2$ algorithm (here $\hat{\mu}^{\pi_E}$). Nevertheless, provided a good representation for the reward function (that is a good choice of basis functions $\phi_i$) and a small estimation error, this should not be a practical problem.

Finally, if our bound relies on the generalization errors $\epsilon_c$ and $\bar{\epsilon}_Q$, the classifier will only use $(\hat{\mu}^{\pi_E}(s_i, a))_{1 \leq i \leq N, a \in \mathcal{A}}$ in the training phase, where $s_i$ are the states from the set $\mathcal{D}$. It outputs $\theta_c$, seen as a reward function, thus the estimated feature expectation $\hat{\mu}^{\pi_E}$ is no longer required. Therefore, practically it should be sufficient to estimate well $\hat{\mu}^{\pi_E}$ on state-action couples $(s_i, a)_{1 \leq i \leq N, a \in \mathcal{A}}$, which allows envisioning Monte-Carlo rollouts for example.

## 4 A Practical Approach

### 4.1 Estimating the Expert Feature Expectation

SCIRL relies on an estimate $\hat{\mu}^{\pi_E}$ of the expert feature expectation. Basically, this is a policy evaluation problem. An already made key observation is that each component of $\mu^{\pi_E}$ is the action-value function of $\pi_E$ for a reward function $\phi_i$: $\mu_i^{\pi_E}(s, a) = Q_{\phi_i}^{\pi_E}(s, a) = [T_{\phi_i}^a v_{\phi_i}^{\pi_E}](s)$. We briefly review its exact computation and possible estimation approaches, and consider possible heuristics.

If the model is known, the feature expectation can be computed explicitly. Let $\Phi \in \mathbb{R}^{|\mathcal{S}| \times p}$ be the feature matrix whose rows contain the feature vectors $\phi(s)^\top$ for all $s \in \mathcal{S}$. For a fixed $a \in A$, let $\boldsymbol{\mu}_a^{\pi_E} \in \mathbb{R}^{|\mathcal{S}| \times p}$ be the feature expectation matrix whose rows are the expert feature vectors, that is $(\mu^{\pi_E}(s, a))^\top$ for any $s \in \mathcal{S}$. With these notations, we have $\boldsymbol{\mu}_a^{\pi_E} = \Phi + \gamma P_a (I - \gamma P_{\pi_E})^{-1} \Phi$. Moreover, the related computational cost is the same order of magnitude as evaluating a single policy (as the costly part, computing $(I - \gamma P_{\pi_E})^{-1}$, is shared by all components).

If the model is unknown, any temporal difference learning algorithm can be used to estimate the expert feature expectation [7], as LSTD (Least-Squares Temporal Differences) [4]. Let $\psi : \mathcal{S} \times \mathcal{A} \to \mathbb{R}^d$ be a feature vector composed of $d$ basis functions $\psi_i \in \mathbb{R}^{\mathcal{S} \times \mathcal{A}}$. Each component $\mu_i^{\pi_E}$ of the

expert feature expectation is parameterized by a vector $\xi_i \in \mathbb{R}^d$: $\mu_i^{\pi_E}(s,a) \approx \xi_i^\top \psi(s,a)$. Assume that we have a training set $\{(s_i, a_i, s'_i, a'_i = \pi_E(s'_i))_{1 \le i \le M}\}$ with actions $a_i$ not necessarily sampled according to policy $\pi_E$ (*e.g.*, this may be obtained by sampling trajectories according to an expert-based $\epsilon$-greedy policy), the aim being to have a better variability of tuples (non-expert actions should be tried). Let $\tilde{\Psi} \in \mathbb{R}^{M \times d}$ (resp. $\tilde{\Psi}'$) be the feature matrix whose rows are the feature vectors $\psi(s_i, a_i)^\top$ (resp. $\psi(s'_i, a'_i)^\top$). Let also $\tilde{\Phi} \in \mathbb{R}^{M \times p}$ be the feature matrix whose rows are the reward's feature vectors $\phi(s_i)^\top$. Finally, let $\Xi = [\xi_1 \ \ldots \ \xi_p] \in \mathbb{R}^{d \times p}$ be the matrix of all parameter vectors. Applying LSTD to each component of the feature expectation gives the LSTD-$\mu$ algorithm [7]: $\Xi = (\tilde{\Psi}^\top(\tilde{\Psi} - \gamma\tilde{\Psi}'))^{-1}\tilde{\Psi}^\top\tilde{\Phi}$ and $\hat{\mu}^{\pi_E}(s,a) = \Xi^\top\psi(s,a)$. As for the exact case, the costly part (computing the inverse matrix) is shared by all feature expectation components, the computational cost is reasonable (same order as LSTD).

Provided a simulator and the ability to sample according to the expert policy, the expert feature expectation may also be estimated using Monte-Carlo rollouts for a given state-action pair (as noted in Sec. 3.2, $\hat{\mu}^{\pi_E}$ need only be known on $(s_i, a)_{1 \le i \le N, a \in \mathcal{A}}$). Assuming that $K$ trajectories are sampled for each required state-action pair, this method would require $KN|\mathcal{A}|$ rollouts.

In order to have a small error $\bar{\epsilon}_Q$, one may learn using transitions whose starting state is sampled according to $\rho_E$ and whose actions are uniformly distributed. However, it may happen that only transitions of the expert are available: $\mathcal{T} = \{(s_i, a_i = \pi_E(s_i), s'_i)_{1 \le i \le N}\}$. If the state-action couples $(s_i, a_i)$ may be used to feed the classifier, the transitions $(s_i, a_i, s'_i)$ are not enough to provide an accurate estimate of the feature expectation. In this case, we can still expect an accurate estimate of $\mu^{\pi_E}(s, \pi_E(s))$, but there is little hope for $\mu^{\pi_E}(s, a \ne \pi_E(s))$. However, one can still rely on some heuristic; this does not fit the analysis of Sec. 3.2, but it can still provide good experimental results, as illustrated in Sec. 6.

We propose such a heuristic. Assume that only data $\mathcal{T}$ is available and that we use it to provide an (accurate) estimate $\hat{\mu}^{\pi_E}(s, \pi_E(s))$ (this basically means estimating a value function instead of an action-value function as described above). We may adopt an optimistic point of view by assuming that applying a non-expert action just delays the effect of the expert action. More formally, we associate to each state $s$ a virtual state $s_v$ for which $p(.|s_v, a) = p(.|s, \pi_E(s))$ for any action $a$ and for which the reward feature expectation is the null vector, $\phi(s_v) = 0$. In this case, we have $\mu^{\pi_E}(s, a \ne \pi_E(s)) = \gamma\mu^{\pi_E}(s, \pi_E(s))$. Applying this idea to the available estimate (recalling that the classifiers only requires evaluating $\hat{\mu}^{\pi_E}$ on $(s_i, a)_{1 \le i \le N, a \in \mathcal{A}}$) provides the proposed heuristic: for $1 \le i \le N$, $\hat{\mu}^{\pi_E}(s_i, a \ne a_i) = \gamma\hat{\mu}^{\pi_E}(s_i, a_i)$.

We may even push this idea further, to get the simpler estimate of the expert feature expectation (but with the weakest guarantees). Assume that the set $\mathcal{T}$ consists of one long trajectory, that is $s'_i = s_{i+1}$ (thus $\mathcal{T} = \{s_1, a_1, s_2, \ldots, s_{N-1}, a_{N-1}, s_N, a_N\}$). We may estimate $\mu^{\pi_E}(s_i, a_i)$ using the single rollout available in the training set and use the proposed heuristic for other actions:

$$\forall 1 \le i \le N, \ \hat{\mu}^{\pi_E}(s_i, a_i) = \sum_{j=i}^{N} \gamma^{j-i}\phi(s_j) \text{ and } \hat{\mu}^{\pi_E}(s_i, a \ne a_i) = \gamma\hat{\mu}^{\pi_E}(s_i, a_i). \quad (5)$$

To sum up, the expert feature expectation may be seen as a vector of action-value functions (for the same policy $\pi_E$ and different reward functions $\phi_i$). Consequently, any action-value function evaluation algorithm may be used to estimate $\mu^\pi(s,a)$. Depending on the available data, one may have to rely on some heuristic to assess the feature expectation for a unexperienced (non-expert) action. Also, this expert feature expectation estimate is only required for training the classifier, so it is sufficient to estimate on state-action couples $(s_i, a)_{1 \le i \le N, a \in \mathcal{A}}$. In any case, estimating $\mu^{\pi_E}$ is not harder than estimating the action-value function of a given policy in the on-policy case, which is much easier than computing an optimal policy for an arbitrary reward function (as required by most of existing IRL algorithms, see Sec. 5).

## 4.2 An Instantiation

As stated before, any MC$^2$ algorithm may be used. Here, we choose the structured large margin approach [18]. Let $\mathcal{L} : \mathcal{S} \times \mathcal{A} \to \mathbb{R}_+$ be a user-defined margin function satisfying $\mathcal{L}(s, \pi_E(s)) \le$

$\mathcal{L}(s, a)$ (here, $\mathcal{L}(s_i, a_i) = 0$ and $\mathcal{L}(s_i, a \neq a_i) = 1$). The MC$^2$ algorithm solves:

$$\min_{\theta, \zeta} \frac{1}{2} \|\theta\|^2 + \frac{\eta}{N} \sum_{i=1}^{N} \zeta_i \quad \text{s.t.} \quad \forall i, \theta^\top \hat{\mu}^{\pi_E}(s_i, a_i) + \zeta_i \geq \max_a \theta^\top \hat{\mu}^{\pi_E}(s_i, a) + \mathcal{L}(s_i, a).$$

Following [13], we express the equivalent hinge-loss form (noting that the slack variables $\zeta_i$ are tight, which allows moving the constraints in the objective function):

$$J(\theta) = \frac{1}{N} \sum_{i=1}^{N} \max_a \theta^\top \hat{\mu}^{\pi_E}(s_i, a) + \mathcal{L}(s_i, a) - \theta^\top \hat{\mu}^{\pi_E}(s_i, a_i) + \frac{\lambda}{2} \|\theta\|^2.$$

This objective function is minimized using a subgradient descent. The expert feature expectation is estimated using the scheme described in Eq. (5).

## 5    Related Works

The notion of IRL has first been introduced in [14] and first been formalized in [11]. A classic approach to IRL, initiated in [1], consists in finding a policy (through some reward function) such that its feature expectation (or more generally some measure of the underlying trajectories' distribution) matches the one of the expert policy. See [10] for a review. Notice that related algorithms are not always able to output a reward function, even if they may make use of IRL as an intermediate step. In such case, they are usually refereed to as apprenticeship learning algorithms.

Closer to our contribution, some approaches also somehow introduce a structure in a classification procedure [8][13]. In [8], a metric induced by the MDP is used to build a kernel which is used in a classification algorithm, showing improvements compared to a non-structured kernel. However, this approach is not an IRL algorithm, and more important assessing the metric of an MDP is a quite involved problem. In [13], a classification algorithm is also used to produce a reward function. However, instead of associating actions to states, as we do, it associates optimal policies (labels) to MDPs (inputs), which is how the structure is incorporated. This involves solving many MDPs.

As far as we know, all IRL algorithms require solving the direct RL problem repeatedly, except [5, 3]. [5] applies to linearly-solvable MDPs (where the control is done by imposing any dynamic to the system). In [3], based on a relative entropy argument, some utility function is maximized using a subgradient ascent. Estimating the subgradient requires sampling trajectories according to the policy being optimal for the current estimated reward. This is avoided thanks to the use of importance sampling. Still, this requires sampling trajectories according to a non-expert policy and the direct problem remains at the core of the approach (even if solving it is avoided).

SCIRL does not require solving the direct problem, just estimating the feature expectation of the expert policy. In other words, instead of solving multiple policy optimization problems, we only solve one policy evaluation problem. This comes with theoretical guarantees (which is not the case of all IRL algorithms, *e.g.* [3]). Moreover, using heuristics which go beyond our analysis, SCIRL may rely solely on data provided by expert trajectories. We demonstrate this empirically in the next section. To the best of our knowledge, no other IRL algorithm can work in such a restrictive case.

## 6    Experiments

We illustrate the proposed approach on a car driving simulator, similar to [1, 16]. The goal si to drive a car on a busy three-lane highway with randomly generated traffic (driving off-road is allowed on both sides). The car can move left and right, accelerate, decelerate and keep a constant speed. The expert optimizes a handcrafted reward $\mathcal{R}_E$ which favours speed, punish off-road, punish collisions even more and is neutral otherwise.

We compare SCIRL as instantiated in Sec. 4.2 to the unstructured classifier (using the same classification algorithm) and to the algorithm of [1] (called here PIRL for Projection IRL). We also consider the optimal behavior according to a randomly sampled reward function as a baseline (using the same reward feature vector as SCIRL and PIRL, the associated parameter vector is randomly sampled).

For SCIRL and PIRL we use a discretization of the state space as the reward feature vector, $\phi \in \mathbb{R}^{729}$: 9 horizontal positions for the user's car, 3 horizontal and 9 vertical positions for the closest

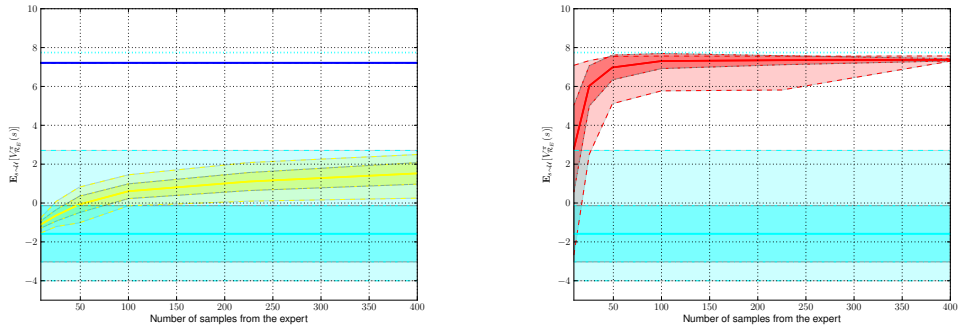

Figure 1: Highway problem. The highest line is the expert value. For each curves, we show the mean (plain line), the standard deviation (dark color) and the min-max values (light color). The policy corresponding to the random reward is in blue, the policy outputted by the classifier is in yellow and the optimal policy according the SCIRL's reward is in red. PIRL is the dark blue line.

traffic's car and 3 speeds. Notice that these features are much less informative than the ones used in [1, 16]. Actually, in [16] features are so informative that sampling a random positive parameter vector $\theta$ already gives an acceptable behavior. The discount factor is $\gamma = 0.9$. The classifier uses the same feature vector reproduced for each action.

SCIRL is fed with $n$ trajectories of length $n$ (started in a random state) with $n$ varying from 3 to 20 (so fed with 9 to 400 transitions). Each experiment is repeated 50 times. The classifier uses the same data. PIRL is an iterative algorithm, each iteration requiring to solve the MDP for some reward function. It is run for 70 iterations, all required objects (a feature expectations for a non-expert policy and an optimal policy according to some reward function at each iteration) are computed exactly using the model. We measure the performance of each approach with $\mathbb{E}_{s \sim \mathcal{U}}[v^\pi_{\mathcal{R}_E}(s)]$, where $\mathcal{U}$ is the uniform distribution (this allows measuring the generalization capability of each approach for states infrequently encountered), $\mathcal{R}_E$ is the expert reward and $\pi$ is one of the following polices: the optimal policy for $\mathcal{R}_E$ (upper baseline), the optimal policy for a random reward (lower baseline), the optimal policy for $\mathcal{R}_{\theta_c}$ (SCIRL), the policy produced by PIRL and the classifier decision rule.

Fig. 1 shows the performance of each approach as a number of used expert transitions (except PIRL which uses the model). We can see that the classifier does not work well on this example. Increasing the number of samples would improve its performance, but after 400 transitions it does not work as well as SCIRL with only a ten of transitions. SCIRL works pretty well here: after only a hundred of transitions it reaches the performance of PIRL, both being close to the expert value. We do not report exact computational times, but running SCIRL one time with 400 transitions is approximately hundred time faster than running PIRL for 70 iteration.

# 7 Conclusion

We have introduced a new way to perform IRL by structuring a linearly parameterized score function-based multi-class classification algorithm with an estimate of the expert feature expectation. This outputs a reward function for which we have shown the expert to be near optimal, provided a small classification error and a good expert feature expectation estimate. How to practically estimate this quantity has been discussed and we have introduced a heuristic for the case where only transitions from the expert are available, along with a specific instantiation of the SCIRL algorithm. We have shown on a car driving simulator benchmark that the proposed approach works well (even combined with the introduced heuristic), much better than the unstructured classifier and as well as a state-of-the-art algorithm making use of the model (and with a much lower computational time). In the future, we plan to deepen the theoretical properties of SCIRL (notably regarding possible heuristics) and to apply it to real-world robotic problems.

**Acknowledgments.** This research was partly funded by the EU FP7 project ILHAIRE (grant n°270780), by the EU INTERREG IVa project ALLEGRO and by the Région Lorraine (France).

## Footnotes

[1]This work can be extended to compact state spaces, up to some technical aspects.

[2]For example, if the Markov chain induced by the expert policy is fast-mixing, sampling a trajectory will quickly lead to sample states according to this distribution.

# References

[1] Pieter Abbeel and Andrew Y. Ng. Apprenticeship learning via inverse reinforcement learning. In *Proceedings of the 21st International Conference on Machine learning (ICML)*, 2004.

[2] Dimitri P. Bertsekas and John N. Tsitsiklis. *Neuro-Dynamic Programming (Optimization and Neural Computation Series, 3)*. Athena Scientific, 1996.

[3] Abdeslam Boularias, Jens Kober, and Jan Peters. Relative entropy inverse reinforcement learning. In *JMLR Workshop and Conference Proceedings Volume 15: AISTATS 2011*, 2011.

[4] Steven J. Bradtke and Andrew G. Barto. Linear Least-Squares algorithms for temporal difference learning. *Machine Learning*, 22(1-3):33–57, 1996.

[5] Krishnamurthy Dvijotham and Emanuel Todorov. Inverse Optimal Control with Linearly-Solvable MDPs. In *Proceedings of the 27th International Conference on Machine Learning (ICML)*, 2010.

[6] Yann Guermeur. VC thoery of large margin multi-category classifiers. *Journal of Machine Learning Research*, 8:2551–2594, 2007.

[7] Edouard Klein, Matthieu Geist, and Olivier Pietquin. Batch, Off-policy and Model-free Apprenticeship Learning. In *Proceedings of the European Workshop on Reinforcement Learning (EWRL)*, 2011.

[8] Francisco S. Melo and Manuel Lopes. Learning from demonstration using MDP induced metrics. In *Proceedings of the European Conference on Machine Learning (ECML)*, 2010.

[9] Rémi Munos. Performance bounds in $L_p$ norm for approximate value iteration. *SIAM journal on control and optimization*, 46(2):541–561, 2007.

[10] Gergely Neu and Czaba Szepesvari. Training Parsers by Inverse Reinforcement Learning. *Machine Learning*, 77(2-3):303–337, 2009.

[11] Andrew Y. Ng and Stuart Russell. Algorithms for Inverse Reinforcement Learning. In *Proceedings of 17th International Conference on Machine Learning (ICML)*, 2000.

[12] Martin L. Puterman. *Markov Decision Processes: Discrete Stochastic Dynamic Programming*. Wiley-Interscience, 1994.

[13] Nathan Ratliff, Andrew D. Bagnell, and Martin Zinkevich. Maximum Margin Planning. In *Proceedings of the 23rd International Conference on Machine Learning (ICML)*, 2006.

[14] Stuart Russell. Learning agents for uncertain environments (extended abstract). In *Proceedings of the 11th annual Conference on Computational Learning Theory (COLT)*, 1998.

[15] Richard S. Sutton and Andrew G. Barto. *Reinforcement Learning: An Introduction*. The MIT Press, 3rd edition, March 1998.

[16] Umar Syed and Robert Schapire. A game-theoretic approach to apprenticeship learning. In *Advances in Neural Information Processing Systems 20 (NIPS)*, 2008.

[17] Csaba Szepesvári. *Algorithms for Reinforcement Learning*. Morgan and Claypool, 2010.

[18] Ben Taskar, Vassil Chatalbashev, Daphne Koller, and Carlos Guestrin. Learning Structured Prediction Models: a Large Margin Approach. In *Proceedings of 22nd International Conference on Machine Learning (ICML)*, 2005.

